# Learning 3D Equivariant Implicit Function with Patch-Level Pose-Invariant Representation

**Xin Hu[1], Xiaole Tang[1], Ruixuan Yu[2], Jian Sun(✉)[1,3]**

[1] Xi'an Jiaotong University, Xi'an, China
[2] Shandong University, Weihai, China
[3] Pazhou Laboratory (Huangpu), Guangzhou, China
`{huxin7020,tangxl}@stu.xjtu.edu.cn,`
`yuruixuan@sdu.edu.cn, jiansun@xjtu.edu.cn`

## Abstract

Implicit neural representation gains popularity in modeling the continuous 3D surface for 3D representation and reconstruction. In this work, we are motivated by the fact that the local 3D patches repeatedly appear on 3D shapes/surfaces if the factor of poses is removed. Based on this observation, we propose the 3D patch-level equivariant implicit function (PEIF) based on the 3D patch-level pose-invariant representation, allowing us to reconstruct 3D surfaces by estimating equivariant displacement vector fields for query points. Specifically, our model is based on the pose-normalized query/patch pairs and enhanced by the proposed intrinsic patch geometry representation, modeling the intrinsic 3D patch geometry feature by learnable multi-head memory banks. Extensive experiments show that our model achieves state-of-the-art performance on multiple surface reconstruction datasets, and also exhibits better generalization to cross-dataset shapes and robustness to arbitrary rotations. Our code will be available at https://github.com/mathXin112/PEIF.git.

## 1 Introduction

Surface reconstruction aims at generating continuous surfaces from discrete point clouds. It is a fundamental and challenging task in current robotics and vision applications [1, 2, 3]. Recently, deep learning-based implicit neural representations (INRs) have emerged as a powerful tool for this task, such as signed distance fields (SDFs) [4, 5], unsigned distance fields (UDFs) [6, 7, 8], and neural vector fields (NVF) [9]. INRs benefit from its continuity, and the ability to handle complicated topology, showing promising performance on surface reconstruction.

Although current INRs-based methods have achieved promising performance in reconstructing surfaces, they suffer from two main challenges. First, most methods [9, 10, 11] deal with the distinct local regions as geometry elements to estimate the query point values, e.g., signed/unsigned distance. However, different local regions may exhibit different poses but with similar intrinsic geometry. The extrinsic poses of these 3D patches prevent the models from capturing the intrinsic geometry of 3D shape patches. Second, INRs [5, 6, 7, 9, 12] without considering equivalence commonly learn the representation of the points using a fixed coordinate frame, implying that if the input points are rotated, the original coordinate mapping may no longer accurately predict the desired output, leading to distortions or inaccuracies. These properties of INRs hinder their applicability to complex 3D scenarios, in particular with regard to their cross-domain generalization ability and robustness to arbitrary transformations like rotations.

To tackle these challenges, we try to eliminate the redundant factor of poses and more focus on the learning of the intrinsic geometric representation of local regions, yielding a patch-level pose-

invariant representation (PPIR) of 3D objects. Based on this representation, we develop a patch-level equivariant implicit function (PEIF), allowing us to achieve the equivariance patch-wisely while effectively encoding arbitrary topology. Specifically, in the PEIF framework, the query/patch pairs are first normalized via a unique pose normalization. Then the query/patch features are extracted and processed via learnable multi-head memory banks to acquire the intrinsic patch geometry representation, which is aggregated with the spatial relation representation, resulting in the patch-level pose-invariant representation. PPIR is then utilized for displacement prediction, which can be proven to be equivariant for $SE(3)$ transformations. These designs enhance the expressive power of INRs with PPIR and enable the PEIF to flexibly adapt to 3D domain gaps as well as arbitrary $SE(3)$ transformations.

Our contributions can be summarized as follows. *First*, we propose a patch-based equivariant implicit function based on the pose-invariant feature learning, facilitating 3D reconstruction robust to 3D shapes $SE(3)$ transformations. *Second*, we design an intrinsic patch geometry representation module encoding rich patch-level pose-invariant features leveraging similar geometric patches. *Third,* the effectiveness of PEIF for surface reconstruction is demonstrated on four datasets including two CAD object datasets, a synthetic scene-level dataset, and a real scan dataset. Experiments show that our method outperforms baseline methods and can effectively reconstruct fine geometric structures, particularly performing well in cross-dataset generalization and the robustness to arbitrary rotations.

## 2   Related Work

### 2.1   Implicit Representation for 3D Shape Reconstruction

Deep learning-based implicit representations have achieved significant advancements, due to their continuity and ability to handle complex geometry structures. Implicit representation for 3D surface reconstruction commonly learns to assign specific values for query points in 3D space. For example, occupancy field (occ) based methods [13, 14, 15, 16, 17, 18] enable the 3D reconstruction as a binary classification problem. The Occupancy Network [14] introduces predictions of spatial point occupancy, while advancements like ConvONet [19] and POCO [10] integrate grid-oriented convolutional or transformer frameworks to enhance performance. Recently, ALTO [20] iteratively refines features from both points and grids, deploying attention-driven interpolation from adjacent grids to decode occupancy values for query points. GridFormer [11] introduces transformer architecture to integrate the advantages of both points and grids for the prediction of occupancy.

SDF/UDF provides a continuous value to each spatial point, indicating the corresponding signed or unsigned distance to the surface. UDF with unsigned distance overcomes the limitations of SDF in handling non-watertight geometries. DeepSDF [4] leverages Multi-Layer Perceptron (MLP) to globally model SDF for entire 3D shape, while DeepLS [5], Instant-NGP [21] and NKSR [22] design more detailed operations to predict the SDF / UDF locally or hierarchically with MLP, kernel function or transformers, etc. GIFS [12] represents general shapes with multi-layer surfaces based on the spatial relationship between points. CAP-UDF [8] employs a field consistency constraint to get consistency-aware UDF. GeoUDF [7] adaptively approximates the UDF and its gradient of a point cloud by leveraging local geometry in a decoupled manner. However, separate learning of UDF values and gradients for points may result in accurate UDF but with the inverted direction problem. To address this issue, NVF [9] proposes an explicit approach to learning implicit representations based on displacement vectors, which ensures both accuracy and correct directional information. In this paper, we adopt this representation, predicting a displacement vector for each point in 3D space. Compared to NVF [9], we design our PEIF over the pose-normalized 3D patches and obtain the $SE(3)$-equivariant implicit function.

### 2.2   $SE(3)$-Equivariant Network

$SE(3)$-equivariance has been extensively studied in both 2D images [23] and 3D point clouds [24, 25, 26, 27, 28]. Given 3D point cloud $X$ and transformation $\forall \zeta \in SE(3)$, a model $f$ is said to be $SE(3)$-equivariant when it satisfies $f \circ \zeta(X) = \zeta \circ f(X)$. Various works have been proposed to achieve $SE(3)$-equivariance based on PCA [29, 30, 25], spherical harmonics [31, 32, 33], equivariant message passing [34, 35, 36], or Vector Neuron [26, 27, 28]. $SE(3)$-equivariant networks are particularly useful for 3D point analysis tasks, such as molecular property or trajectory modeling [34, 35, 36, 37], protein structure prediction [38, 39, 40], 3D shape recognition [26, 27, 28], and robotics [41, 42, 43].

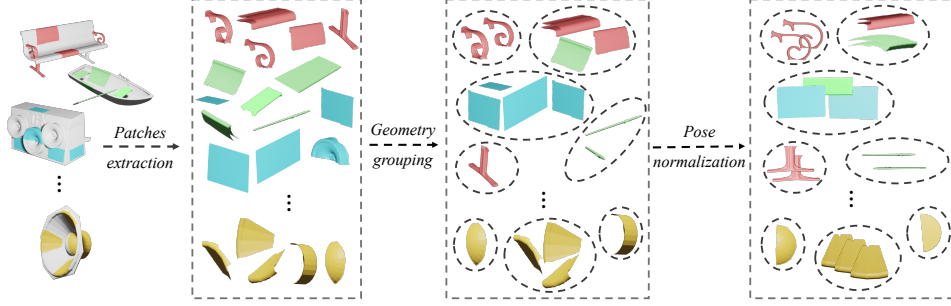

Figure 1: Local 3D patches may exhibit geometric similarity, but with different poses. When the pose is removed, these local regions appear repeatedly.

Introducing $SE(3)$-equivariance to build an orientation-robust implicit field is one of the motivations of this work. There are few works involving equivariance in the implicit field. EFEM [44] uses Vector Neuron [27] to learn equivariant shape representations before shape segmentation. E-GraphONet [24] utilizes basic Vector Neuron [27] layers to design graph networks, achieving locally $SO(3)$-invariant features for implicit function learning. E-GraphONet [24] is the most related work to ours, which extends neurons from 1D scalars to 3D vectors for each point. In comparison, our PEIF employs lightweight PCA to achieve pose-invariant patch-level representation and leverages a multi-head memory bank for intrinsic geometry representation, achieving state-of-the-art 3D reconstruction performance.

## 3 Problem Statement for Equivariant Neural Vector Field

In this section, we first introduce the implicit representation, namely the neural vector fields (NVF) [9], and then introduce the equivariant implicit function of this representation.

Given a sparse point cloud $X \in \mathbb{R}^{N_x \times 3}$ sampled on a shape $\mathcal{X}$, and a query set $Q \in \mathbb{R}^{N_q \times 3}$ sampled near the surface of $\mathcal{X}$, where $N_x$ and $N_q$ represent the number of input points and query points respectively. A shape $\mathcal{X}$ is defined as the zero displacement of the implicit function $\mathcal{F}$

$$\mathcal{X} = \left\{ x \in \mathbb{R}^3 \,\middle|\, \mathcal{F}(x) = \vec{0} \right\}, \tag{1}$$

where $x$ is a point in point cloud $X$, containing its spatial coordinate. $\vec{0}$ represents the zero displacement of the point $x$. For a query point $q \in \mathbb{R}^3$, the implicit function $\mathcal{F}$ is formulated by

$$\mathcal{F}(q) = \Delta q = \hat{x} - q, \quad \text{where } \hat{x} = \text{argmin}_{x \in \mathcal{X}} \|x - q\|, \tag{2}$$

and $\hat{x}$ is the nearest point of query $q$ on the $\mathcal{X}$.

**Definition 1** (Equivariant Implicit Function). *Given an abstract group $G$, the implicit function $\mathcal{F}$ based on NVF is equivariant with regard to $G$, if*

$$\mathcal{F}(\zeta \circ q) = \zeta \circ (\mathcal{F}(q)) = \Delta q, \quad \forall \zeta \in G, \tag{3}$$

where $q$ is a query point near or on the surface of shape $\mathcal{X}$. In this work, the group $G$ is $SE(3)$.

## 4 Equivariant Neural Implicit Function

In this work, we aim to develop an equivariant implicit function model grounded on neural vector field representations. We achieve this goal by firstly learning patch-level pose-invariant representation (PPIR), and then designing shape-level equivariant implicit representation. The overview of our method is introduced in Section 4.1, with the detailed designs presented in Sections 4.2 and 4.3.

### 4.1 Overview of the Basic Idea

Given point cloud $X$, implicit representations conventionally involve sampling a set of query points $Q$ and employing an implicit function $\mathcal{F}$ to compute their associated implicit values. Typically, the

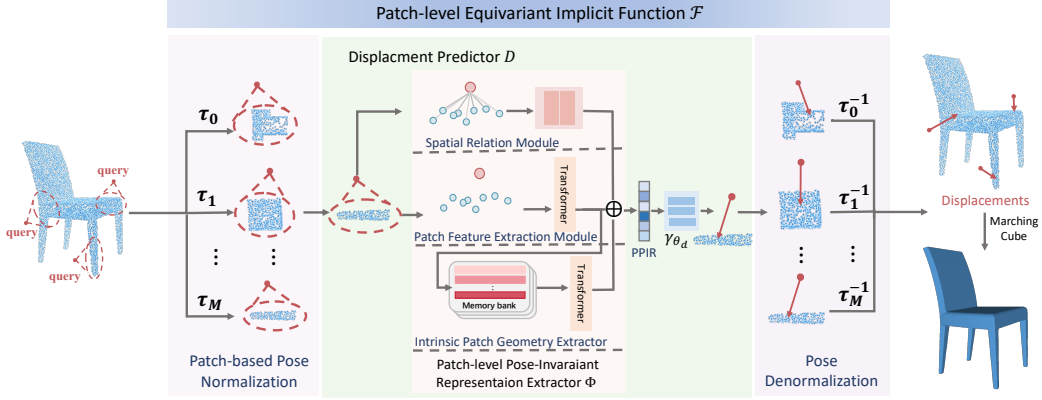

Figure 2: Overview of the proposed PEIF. Given query points, the local patches are selected using KNN. The query/patch pairs are normalized by pose transformations $\tau$. The displacements of query points to the surface are predicted by displacement predictor $D$. The implicit function is equivariant under the $SE(3)$ transformations of the input. Finally, the mesh is generated by marching cubes [1] algorithm.

depiction of a query point $q \in Q$ depends on its K-nearest neighbors (KNN) in $X$. As shown in Figure 1, it is observed that some local KNN patches exhibit identical geometric structures if ignoring their pose variations in $SE(3)$, and the local patches across 3D objects also repeatedly appear. Based on this observation, we design an equivariant implicit function based on patch-level pose-invariant representation, capturing recurring geometric patterns invariant to pose transformation.

Before delving into the specific details of our approach, we present the overall framework as shown in Figure 2. Given query set $Q = \{q_i\}$, the corresponding patch for $q_i$ on point cloud $X$ is $P_i = \{p_{i,k}\}_{k=0}^{K}$, *i.e.*, the KNN of $q_i$ based on Euclidean distance. The point patch $P_i$ and query point $q_i$ are firstly normalized by patch-based pose normalization $\tau_i$, achieving invariant ones under $SE(3)$ transformation of patch $P_i$. We then feed $\{\tau_i(P_i), \tau_i(q_i)\}$ to the displacement predictor $D$ for $SE(3)$-invariant representation learning and displacement prediction. Finally, this predicted displacement is transformed back to the pose of $P_i$ with $\tau_i^{-1}$. The overall displacement prediction can be written as

$$\Delta q_i = \mathcal{F}(q_i) = \tau_i^{-1} \circ D \circ \{\tau_i(P_i), \tau_i(q_i)\}. \tag{4}$$

This framework is $SE(3)$-equivariant for patch $P_i$ and point cloud $X$. The detailed design of the patch-based pose-normalization $\tau$ and displacement predictor $D$ are presented in the following Sections 4.2 and 4.3 respectively. We remove index $i$ for brevity and denote the query point, point patch, and pose-normalization as $q \in \mathbb{R}^3$, $P \in \mathbb{R}^{K \times 3}$ and $\tau$ respectively in the following paragraphs.

## 4.2 Pose Normalization

Geometrically identical patches are expected to maintain consistency across various pose transformations, enabling their representations to complement and reinforce each other. Accordingly, we employ Principal Component Analysis (PCA) to extract the pose-invariant information for patch $P$.

We first decenter the patch $P$ by subtracting the points center $\mu$, then obtain the rotation matrix $U$ by computing the Singular Value Decomposition (SVD) [45] over the covariance matrix $(P-\mu)^\top (P-\mu)$. The pose-normalized patch $\bar{P}$ and query point $\bar{q}$ are derived as

$$\bar{P} \triangleq \tau(P) = (P - \mu)U, \quad \bar{q} \triangleq \tau(q) = (q - \mu)U. \tag{5}$$

The pose-normalized patch $\bar{P}$ and query point $\bar{q}$ are invariant under $SE(3)$ transformation of $P$, and we take them as input to our displacement predictor $D$. The prediction $D \circ \{\tau(P), \tau(q)\}$ is also invariant as proven in following Lemma 1. Note that we uniquely determine $U$ as [46] to solve the direction uncertainty problem brought by PCA.

**Lemma 1** *With $\tau$ as our pose normalization, and $D$ as displacement predictor, $D \circ \{\tau(P), \tau(q)\}$ is invariant under $SE(3)$ transformation of $P$.*

Please refer to the Appendix for proof. The displacement predictor will be introduced as follows.

### 4.3 Displacement Predictor Design on Normalized Patches

Taking the pose-normalized patch $\bar{P}$ and query $\bar{q}$ as input, the displacement predictor $D$ is designed to predict the displacement $\Delta\bar{q}$. As shown in Figure 2, predictor $D$ comprises a pose-invariant feature extractor $\Phi$ and a MLP $\gamma_{\theta_d}$. Specifically, the feature extractor $\Phi$ is composed of three modules: the Spatial Relation Module (SRM) for query point feature learning, which models the spatial relative relationship between $\bar{q}$ and $\bar{P}$; the Patch Feature Extraction Module (PFEM) for patch feature learning, which extracts patch feature leveraging correlation in feature space; the Intrinsic Patch Geometry Extractor (IPGE), which learns memory-augmented patch representation.

**Spatial Relation Module.** We design SRM to learn query point features based on spatial relation within query $\bar{q}$ and patch $\bar{P} = \{\bar{p}_i\}_{i=1}^K$. Specifically, the point-wise representation $z_i$ of point $\bar{p}_i \in P$ is firstly computed as

$$z_i = \gamma_{\theta_s}(\bar{p}_i, \bar{p}_i - \bar{q}), \quad i = 1, 2, \ldots, K, \tag{6}$$

$\gamma_{\theta_s}(\cdot)$ is set as MLP. Taking query point position and relative offset as inputs, $z_i$ is expected to directly capture the geometric patterns. Then we aggregate $z_i$ with simple concatenation operator $\oplus$ by

$$h_{\bar{q}} = z_1 \oplus \cdots \oplus z_K. \tag{7}$$

Feature $h_{\bar{q}} \in \mathbb{R}^{K \times D}$ contains the relative feature of query point $\bar{q}$ to the patch $\bar{P}$. We take it as a representation for the query point $\bar{q}$.

**Patch Feature Extraction Module.** We further design PFEM to learn the patch feature for $\bar{P}$. Taking the point positions of $\bar{q}$ and $\bar{P} = \{\bar{p}_i\}_{i=1}^K$ as inputs, we first lift them from Euclidean space to feature space via two MLPs $\gamma_{\theta_p}(\cdot)$ and $\gamma_{\theta_q}(\cdot)$ as

$$f_{\bar{q}} = \gamma_{\theta_q}(\bar{q}), \quad f_{\bar{p}_i} = \gamma_{\theta_p}(\bar{p}_i), \quad i = 1, 2, \ldots, K, \tag{8}$$

where $f_{\bar{q}}, f_{\bar{p}_i} \in \mathbb{R}^{1 \times D}$ are the learned point-wise features. Then, a transformer is designed to obtain the patch feature, by encoding the feature attention between point $\bar{p}_i$ and query point $\bar{q}$ as

$$f_{\bar{P}_w} \triangleq \sum_{i=1}^K a_i \cdot (f_{\bar{p}_i} W_V), \quad \text{where } \{a_i\}_{i=1}^K = \text{Softmax}\left(\{(f_{\bar{q}} W_Q)(f_{\bar{p}_i} W_O)^\top\}\right), \tag{9}$$

where $a_i$ represents the attention score between query point $\bar{q}$ and patch points $\bar{p}_i$. The matrices $W_Q, W_O, W_V \in \mathbb{R}^{D \times D}$ are learnable parameters. Patch feature $f_{\bar{P}_w}$ is aggregated from all the points features in patch $P$, while different patches may have diverse point distributions. To mitigate the effects of point density in patches, we further design the importance-aware patch feature $f_{\bar{P}_s}$ by selecting the top-$K_d$ important points, and aggregating their features as

$$f_{\bar{P}_s} \triangleq \sum_{i=1}^{K_d} b_i f_{\bar{p}_i}, \tag{10}$$

where $\{b_i\}_{i=1}^{K_d}$ is the selected top-$K_d$ attention scores from $\{a_i\}_{i=1}^K$. The final patch feature $f_{\bar{P}}$ from this PFEM is

$$f_{\bar{P}} = \lambda_1 f_{\bar{P}_w} + \lambda_2 f_{\bar{P}_s}, \tag{11}$$

where $\lambda_1, \lambda_2$ are learnable combination coefficients.

**Intrinsic Patch Geometry Extractor.** As discussed in Section 4.1, the normalized point patches can be grouped into different geometric patterns across patches or shapes. To learn the intrinsic features hidden behind those geometric patterns, we propose IPGE to enhance the patch features. Specifically, a learnable multi-head memory bank $\mathcal{M} = \{\mathcal{M}_i\}_{i=1}^{N_M}$ is constructed, and it is shared across the whole dataset to implicitly model the patch patterns. Then the patch feature $f_{\bar{P}}$ is enhanced by querying and aggregating each memory item as

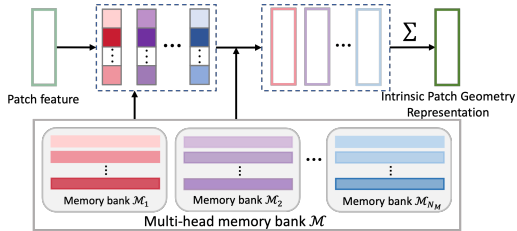

Figure 3: Feature enhancement with IPGE.

$$g_{\bar{P}} = \sum_{i=1}^{N_M} w_i \mathcal{M}_i, \quad w_i = \text{Softmax}(f_{\bar{P}} \mathcal{M}_i^\top). \tag{12}$$

The memory weight $w_i$ is defined as softmax-normalized similarity vectors between query feature $f_{\bar{P}}$ and the entries of $\mathcal{M}_i \in \mathbb{R}^{C \times D}$. Figure 3 illustrates the procedures of this module.

**Displacement Prediction.** Based on query point features $\{h_{\bar{q}}, f_{\bar{q}}\}$, point-wise patch feature $\{f_{\bar{p}_i}\}_{i=1}^{K}$, enhanced patch feature $g_{\bar{P}}$, the patch-level pose-invariant (PPIR) representation is achieved by

$$f_{PPIR} = \gamma_{\theta_a} \left( h_{\bar{q}} \oplus f_{\bar{q}} \oplus \{f_{\bar{p}_i}\} \right) \oplus g_{\bar{P}}. \tag{13}$$

Then the displacement for query point $\bar{q}$ can then be derived by

$$\Delta\bar{q} = \gamma_{\theta_d} \left( f_{PPIR} \right). \tag{14}$$

Both $\gamma_{\theta_d}$ and $\gamma_{\theta_a}$ are set as MLPs. Finally, $\Delta\bar{q}$ is transformed back with pose denormalization, *i.e.*, the inverse transformation of $\tau$, achieving the final $SE(3)$-equivariant displacement estimation $\Delta q = \tau^{-1}(\Delta\bar{q})$. The $SE(3)$-invariance of $f_{PPIR}$ can be found in Lemma 1, and the $SE(3)$-equivariance of learned implicit representation can be found in the following Theorem 1, please refer to Appendix for proof.

**Theorem 1** *Given query point $q$ and patch $P$, implicit function $\mathcal{F}(q)$ is $SE(3)$-equivariant.*

### 4.4 Network Training and Inference

Sections 4.2 and 4.3 illustrate how to obtain the displacement for a query point, where the trained parameters include the parameters $\theta_s, \theta_p, \theta_q, \theta_a, \theta_d$ of five MLPs, the multi-head memory bank $\mathcal{M}$ and parameters $\lambda_1, \lambda_2$. To optimize the implicit function $\mathcal{F}$, we design a joint loss function over the query set $Q$ to train our method in an end-to-end manner.

**Displacement Optimization Loss.** We compute $L_1$-loss between the predicted displacement $\Delta q_i$ for each query point $q_i \in Q$ and its ground-truth displacement $\Delta\hat{q}_i$ as

$$\mathcal{L}_d = \frac{1}{N_q} \sum_{i=1}^{N_q} |\Delta q_i - \Delta\hat{q}_i|. \tag{15}$$

**Patch Discrimination Loss.** The items in memory should be apart from each other to enhance the representativeness of the memory items. To ensure this, we design the patch discrimination loss as

$$\mathcal{L}_m = \sum_{i=1}^{N_M} \sum_{m \neq m'}^{N_{M_i}} \frac{\max(\langle \mathcal{M}_i[m], \mathcal{M}_i[m'] \rangle, 0)}{N_{M_i}(N_{M_i} - 1)}, \tag{16}$$

which is similar to cosine embedding loss [47] with a margin set to 0. $N_{M_i}$ is the number of the items in memory bank $\mathcal{M}_i$. The overall loss function is finally written as $\mathcal{L} = \mathcal{L}_d + \beta\mathcal{L}_m$, where $\beta$ is a hyper-parameter for balancing the two terms.

**3D Reconstruction in Inference Stage.** We employ the Marching Cubes (MC) algorithm proposed by MeshUDF [48], which can reconstruct surfaces on UDFs. We first discretize the 3D volume into a 3D grid with a resolution of $N_R$, resulting in $N_R^3$ grid points as the query set $Q$. Then, we use the implicit function $\mathcal{F}$ to predict the displacement $\Delta q$ of each query point $q$. Similar to [9], we get the UDF value and gradient of $q$ as $d = \|\Delta q\|_2$ and $\nabla_q = \frac{\Delta q}{\|\Delta q\|_2}$. Based on $d$ and $\nabla_q$, the MC in MeshUDF [48] can reconstruct the surface of the input point cloud as mesh.

## 5 Experiments

**Implementation Details.** We implement our PEIF in Pytorch [49] using Adam optimizer [50]. The learning rate is $8 \times 10^{-4}$. For each query point, the size of the neighborhood is set as $K = 32$ for ShapeNet [51] and ABC [52] datasets, $K = 54$ for Synthetic Rooms [19] dataset. We set $\beta = 0.1$ in the training loss and $N_m = 4$ for the memory bank. Please refer to the Appendix for details on the structures of involved MLPs, and the effect of different values of $\beta$. We conducted all experiments on one NVIDIA RTX 4090 GPU.

**Datasets.** We experiment on four datasets including ShapeNet [51], ABC [52], Synthetic Rooms [19], MGN [53]. (1) ***ShapeNet*** [51], as pre-processed by [7], contains watertight meshes of shapes in

Table 1: The reconstruction results of ShapeNet [51]. All models are trained on the base classes and evaluated on both the base classes and novel classes. Note that E-GraphONet is the equivariant version of GraphONet.

| Method | Base | | | | Novel | | | |
|---|---|---|---|---|---|---|---|---|
| | CD ↓ | EMD ↓ | NC ↑ | F-Score ↑ | CD ↓ | EMD ↓ | NC ↑ | F-Score ↑ |
| POCO [10] | 0.395 | 3.937 | 0.929 | 0.970 | 0.520 | 4.941 | 0.906 | 0.954 |
| GIFS [12] | 0.385 | 3.859 | 0.932 | 0.962 | 0.422 | 4.923 | 0.917 | 0.942 |
| ALTO [20] | 0.352 | 3.851 | 0.930 | 0.963 | 0.357 | 4.924 | 0.920 | 0.929 |
| NVF [9] | 0.255 | 3.766 | 0.938 | 0.983 | 0.266 | **4.721** | 0.921 | 0.982 |
| GeoUDF [7] | 0.225 | 3.761 | **0.957** | 0.997 | 0.219 | 4.735 | 0.934 | **0.997** |
| GridFormer [11] | 0.284 | 3.768 | 0.942 | 0.984 | 0.289 | 4.725 | 0.928 | 0.985 |
| GraphONet [24] | 0.389 | 3.868 | 0.921 | 0.932 | 0.461 | 4.733 | 0.917 | 0.952 |
| E-GraphONet [24] | 0.479 | 3.834 | 0.917 | 0.927 | 0.508 | 4.743 | 0.911 | 0.942 |
| PEIF(Ours) | **0.215** | **3.755** | 0.956 | **0.998** | **0.209** | 4.725 | **0.941** | **0.997** |

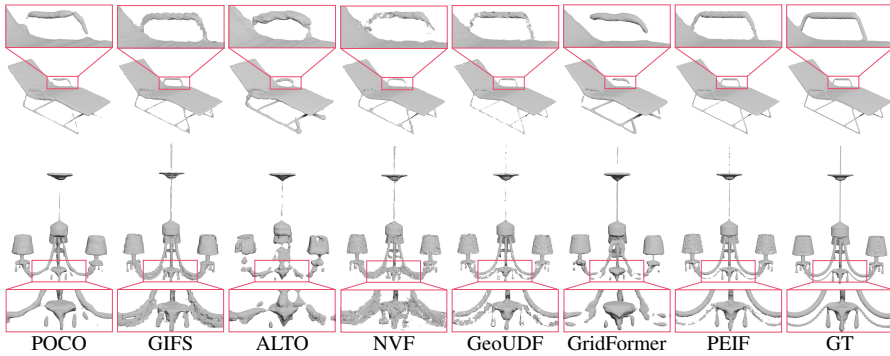

POCO   GIFS   ALTO   NVF   GeoUDF   GridFormer   PEIF   GT

Figure 4: The qualitative results of ShapeNet [51] dataset. The object is selected from meshes used for class-unseen reconstruction (novel classes in Table 1).

13 classes. Following the experimental setting in [9], we select *cars, chairs, planes, and tables* as base classes in Table 1, and *speakers, bench, lamps, and watercraft* as novel classes in Table 1 for category-unseen reconstruction, only for testing. (2) ***ABC*** [52] has one million CAD models, mainly mechanical objects. We use the splits from [54] and select watertight meshes for experiments: 3599/883/98 shapes for training/validation/testing. (3) ***Synthetic Rooms*** [19] contains 5k synthetic room scenes composed of random walls, floors, and ShapeNet objects. We adopt the same train/validation/test division in [19]. (4) ***MGN*** [53] is a real scanned dataset containing 5 clothing categories. To generate watertight surfaces, we employ the method [55] for preprocessing. Specifically, we sample 3k points on the surface as input points for ShapeNet and ABC datasets, while 10k input points for Synthetic Rooms. Then 2048 query points are sampled near the surface for ShapeNet, ABC, and Synthetic Rooms. All experiments are tested on 10k points.

**Evaluation Metrics.** We use the Chamfer-L1 distance (CD, $\times 10^{-2}$), Earth Mover Distance (EMD, $\times 10^{-2}$), Normal Consistency (NC), and F-Score (with threshold value 1%) metrics for our evaluation.

**Baselines.** To evaluate the effectiveness of our methods, baselines used for comparison include the equivariant network E-GraphONet [24], and six non-equivariant networks, including POCO [10], GIFS [12], ALTO [20], NVF [9], GeoUDF [7], GridFormer[11]. For fairness, we trained these networks from scratch under the same training/validation/testing dataset splitting.

## 5.1 Results and Comparisons

**3D Object Datasets Reconstruction.** We first report the results of the 3D object reconstruction on the object datasets: ShapeNet [51] and ABC [52]. The quantitative results on base and novel classes of ShapeNet [51] are shown in Table 1, our PEIF achieves better results both in base and novel classes, especially in terms of the CD and F-Score metrics. Qualitative comparisons are provided in Figure 4. Compared with other competitors, our method can capture fine-grained details, and the

Table 2: Comparison of different methods on ABC [52] and Synthetic Rooms [19] datasets.

| Method | ABC | | | | SyntheticRoom | | | |
|---|---|---|---|---|---|---|---|---|
| | CD ↓ | EMD ↓ | NC ↑ | F-Score ↑ | CD ↓ | EMD ↓ | NC ↑ | F-Score ↑ |
| POCO [10] | 0.475 | 2.785 | 0.957 | 0.941 | 0.512 | 2.624 | 0.896 | 0.973 |
| GIFS [12] | 0.339 | 2.765 | 0.950 | 0.985 | 0.425 | 2.658 | 0.913 | 0.984 |
| ALTO [20] | 0.451 | 2.739 | 0.943 | 0.958 | 0.492 | 2.426 | 0.901 | 0.975 |
| NVF [9] | 0.245 | 2.685 | 0.963 | 0.996 | 0.504 | 2.052 | 0.925 | 0.979 |
| GeoUDF [7] | 0.245 | 2.688 | 0.964 | 0.997 | 0.383 | 2.182 | 0.921 | 0.988 |
| GridFormer [11] | 0.299 | **2.662** | 0.964 | 0.981 | 0.465 | 2.252 | 0.913 | 0.978 |
| E-GraphONet [24] | 0.432 | 2.688 | 0.910 | 0.906 | 0.485 | 2.534 | 0.903 | 0.980 |
| PEIF (Ours) | **0.241** | 2.672 | **0.969** | **0.998** | **0.314** | **2.045** | **0.925** | **0.994** |

overall topology of the shape is more consistent. Additional instances are provided in the Appendix (Figure 8). We further evaluate the compared methods on the ABC [52] dataset. The quantitative results in Table 2 demonstrate that our PEIF achieves competitive performance compared to both equivariant and non-equivariant methods. Visualizations are provided in Figure 9 of the Appendix.

**3D Scene Datasets Reconstruction.** Table 2 shows the quantitative results on the Synthetic Rooms dataset. Our PEIF shows state-of-the-art performance under all quantitative metrics. The competitive competitors such as GridFormer [11] and POCO [10] produce smooth but incomplete surfaces. Other methods like GeoUDF [7] and NVF [9] produce results with rough surfaces. In contrast, our method reconstructs relatively smooth surfaces with fewer issues of completeness and consistency. Visualizations are provided in Figure 10 of the Appendix.

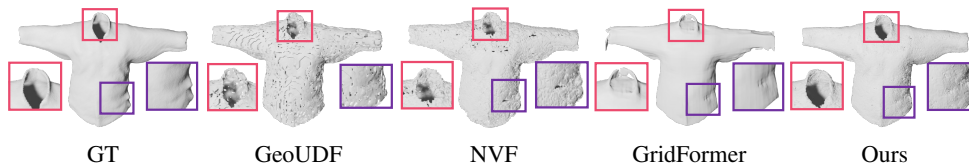

| GT | GeoUDF | NVF | GridFormer | Ours |

Figure 5: The visual example of cross-domain evaluation on the real scanned dataset MGN [53], where the model is pre-trained on Synthetic Rooms dataset [19].

**Cross-domain Evaluation on Real-world Dataset.** We test and compare our method with the state-of-the-art methods NVF, GoeUDF, and GridFormer on MGN [53] dataset using the trained models on Synthetic Rooms [19]. Table 3 shows that the compared methods generally exhibit a declined performance in the presence of a synthetic-real domain gap.

Table 3: The cross-domain evaluation on MNG dataset [53].

| Method | CD ↓ | EMD ↓ | NC ↑ | F-Score ↑ |
|---|---|---|---|---|
| NVF [9] | 0.272 | 4.329 | 0.847 | 0.991 |
| GeoUDF [7] | 0.249 | 4.269 | 0.891 | 0.995 |
| GridFormer [11] | 0.281 | 4.675 | 0.916 | 0.969 |
| E-GraphONet [24] | 0.433 | 3.817 | 0.863 | 0.920 |
| PEIF (Ours) | **0.241** | **2.672** | **0.969** | **0.998** |

However, in the presence of such a domain gap, our PEIF still achieves notable performance under all metrics. Figure 5 displays the visual comparison. The competitors either produce a rough surface or suffer from shape incompleteness. As a comparison, our PEIF reconstructs a complete surface with fine-grained details. These results show that our PEIF trained on the synthetic data can be well generalized to real scenarios. The reason might be that the pose-normalized patches are the basic elements for composing different shapes, and our PEIF is based on the pose-invariant patch representations.

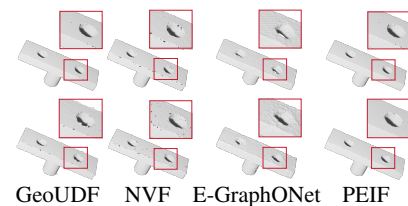

| GeoUDF | NVF | E-GraphONet | PEIF |

Figure 6: Visual results before (top) and after (down) arbitrary rotations.

Table 4: Performance under Arbitrary $SO(3)$ rotations.

| | Methods | CD ↓ | EMD ↓ | NC ↑ | F-Score ↑ |
|---|---|---|---|---|---|
| w/o rotations | NVF [9] | 0.245 | 2.685 | 0.963 | 0.997 |
| | GeoUDF [7] | 0.245 | 2.688 | 0.964 | 0.997 |
| | E-GraphONet [24] | 0.432 | 2.688 | 0.910 | 0.906 |
| | PEIF (Ours) | 0.241 | 2.672 | 0.969 | 0.998 |
| w/ rotations | NVF [9] | 0.261 | 2.699 | 0.946 | 0.993 |
| | GeoUDF [7] | 0.253 | 2.698 | 0.957 | 0.994 |
| | E-GraphONet [24] | 0.433 | 2.689 | 0.912 | 0.909 |
| | PEIF (Ours) | 0.241 | 2.675 | 0.968 | 0.998 |

Table 5: Ablation study on ABC [52] dataset.

| | Setting | CD ↓ | EMD ↓ | NC ↑ | F-Score ↑ |
|---|---|---|---|---|---|
| Pose normalization | w/o | 0.261 | 2.677 | 0.933 | 0.993 |
| Memory bank | $N_M = 0$ | 0.275 | 2.713 | 0.957 | 0.985 |
| | $N_M = 1$ | 0.244 | 2.705 | 0.962 | 0.997 |
| | $N_M = 2$ | 0.244 | 2.696 | 0.962 | 0.997 |
| | $N_M = 3$ | 0.243 | 2.694 | 0.964 | 0.998 |
| | $N_M = 5$ | 0.244 | 2.691 | 0.961 | 0.997 |
| KNN | $K = 54$ | 0.245 | 2.692 | 0.962 | 0.996 |
| | $K = 20$ | 0.249 | 2.691 | 0.957 | 0.996 |
| Full Model ($N_M = 4$, and $K = 32$) | | 0.241 | 2.672 | 0.969 | 0.998 |

Table 6: Model size and inference time.

| Method | # Para (M) | Time (s) |
|---|---|---|
| POCO [10] | 12.19 | 45.10 |
| GIFS [12] | 3.51 | 16.68 |
| ALTO [20] | 2.64 | 20.78 |
| NVF [9] | 10.29 | 73.90 |
| GeoUDF [7] | 0.74 | 124.73 |
| GridFormer [11] | 4.11 | 13.32 |
| GraphONet [24] | 0.06 | 1.72 |
| E-GraphONet [24] | 0.07 | 1.92 |
| Ours | 7.65 | 40.98 |

**Robustness to rotations.** We compare the robustness of two equivariant networks (PEIF and E-GraphONet), and two non-equivariant networks (NVF and GeoUDF) to arbitrary rotations. All methods are trained with canonical pose and tested with arbitrary rotations on the ABC [52] dataset. The quantitative and qualitative results are reported in Table 4 and Figure 5, respectively. In Table 4, "w/ rotation" and "w/o rotation" represent that the testing input point cloud is with and without arbitrary rotation, respectively. The visual results presented in Figure 5 illustrate that our PEIF can retain stable performance under arbitrary rotations, which is consistent with the numerical results presented in Table 4. These results justify the robustness of our PEIF to arbitrary rotations.

## 5.2 Ablation Study and Model Analysis

We conduct ablation studies and present the model size and inference time on the ABC [52] dataset.

**Effect of Pose Normalization.** As shown in the 2nd row in Table 5, after removing the patch-level pose normalization, all metrics decline. Particularly, the NC metric is notably affected.

**Effect of the Multi-head Memory Bank.** As demonstrated in Table 5, the performance of our PEIF deteriorates significantly when the multi-head memory bank, *i.e.*, the intrinsic patch geometry extractor, is removed. The model performs better with $N_M$ increase from 1 to 4. When $N_M = 5$, the performance of the model starts to deteriorate.

**Number of Neighbour Points $K$.** The size of KNN determines the number of points in each patch. We report the performance of our PEIF with different patch sizes in Table 5. The results show that our method is relatively stable to the size of KNN.

**Model Size and Computational Time.** We compare model size and inference time on the ABC [52] dataset. In Table 6, our network is comparable to other methods in the number of parameters. The computation time of our approach for 3D reconstruction of one point cloud is lower than the state-of-the-art models NVF, and GeoUDF, but higher than GrridFormer and E-GraphONet. However, our method achieves the best accuracy for 3D reconstruction as shown in experiments.

## 6 Conclusion

In this work, we propose a patch-level equivariant implicit function, based on the patch-level pose invariant feature representation over the pose-normalized query points and corresponding neighboring patches. The proposed representation achieves promising results in 3D reconstruction both quantitatively and qualitatively, and generates shapes with better geometry details and robustness to $SE(3)$ transforms. Due to the flexibility of the patch-based representation, in the future, we plan to extend this approach to larger-scale 3D reconstruction of real scans. Additionally, our patch-based pose invariant representation can be taken as a foundation network for pre-training, followed by fine-tuning on few-shot examples.

**Limitation.** As an implicit network, one limitation is that PEIF relies on the query points and estimating the displacement point-wisely. For scaling up to a larger scale, we plan to utilize a multi-scale technique and importance sampling of query points for efficient displacement field estimation.

**Impact Statement.** This work aims to advance the field of equivariant deep learning with applications in 3D surface reconstruction. It may be valuable to the research of equivariant implicit neural representation and geometric modeling of 3D shapes and has no ethical concerns as far as we know.

## Acknowledgments

This work was supported by the National Key R&D Program 2021YFA1003002, NSFC 12125104, U20B2075, 12326615, 62306167, Shandong Province Natural Science Foundation ZR2024QA161.

## References

[1] William E Lorensen and Harvey E Cline. Marching cubes: A high resolution 3d surface construction algorithm. In *Seminal graphics: pioneering efforts that shaped the field*, pages 347–353. 1998.

[2] Hugues Hoppe. Poisson surface reconstruction and its applications. In *Proceedings of the 2008 ACM symposium on Solid and physical modeling*, pages 10–10, 2008.

[3] Philipp Mittendorfer and Gordon Cheng. 3d surface reconstruction for robotic body parts with artificial skins. In *2012 IEEE/RSJ International Conference on Intelligent Robots and Systems*, pages 4505–4510. IEEE, 2012.

[4] Jeong Joon Park, Peter Florence, Julian Straub, Richard Newcombe, and Steven Lovegrove. Deepsdf: Learning continuous signed distance functions for shape representation. In *Proceedings of the IEEE/CVF Conference on Computer Vision and Pattern Recognition*, pages 165–174, 2019.

[5] Rohan Chabra, Jan E Lenssen, Eddy Ilg, Tanner Schmidt, Julian Straub, Steven Lovegrove, and Richard Newcombe. Deep local shapes: Learning local sdf priors for detailed 3d reconstruction. In *Computer Vision–ECCV 2020: 16th European Conference, Glasgow, UK, August 23–28, 2020, Proceedings, Part XXIX 16*, pages 608–625, 2020.

[6] Julian Chibane, Gerard Pons-Moll, et al. Neural unsigned distance fields for implicit function learning. *Advances in Neural Information Processing Systems*, 33:21638–21652, 2020.

[7] Siyu Ren, Junhui Hou, Xiaodong Chen, Ying He, and Wenping Wang. Geoudf: Surface reconstruction from 3d point clouds via geometry-guided distance representation. In *Proceedings of the IEEE/CVF International Conference on Computer Vision*, pages 14214–14224, 2023.

[8] Junsheng Zhou, Baorui Ma, Yu-Shen Liu, Yi Fang, and Zhizhong Han. Learning consistency-aware unsigned distance functions progressively from raw point clouds. *Advances in Neural Information Processing Systems*, 35:16481–16494, 2022.

[9] Xianghui Yang, Guosheng Lin, Zhenghao Chen, and Luping Zhou. Neural vector fields: Implicit representation by explicit learning. In *Proceedings of the IEEE/CVF Conference on Computer Vision and Pattern Recognition*, pages 16727–16738, 2023.

[10] Alexandre Boulch and Renaud Marlet. Poco: Point convolution for surface reconstruction. In *Proceedings of the IEEE/CVF Conference on Computer Vision and Pattern Recognition*, pages 6302–6314, 2022.

[11] Shengtao Li, Ge Gao, Yudong Liu, Yu-Shen Liu, and Ming Gu. Gridformer: Point-grid transformer for surface reconstruction. *arXiv preprint arXiv:2401.02292*, 2024.

[12] Jianglong Ye, Yuntao Chen, Naiyan Wang, and Xiaolong Wang. Gifs: Neural implicit function for general shape representation. In *Proceedings of the IEEE/CVF Conference on Computer Vision and Pattern Recognition*, pages 12829–12839, 2022.

[13] Zhiqin Chen and Hao Zhang. Learning implicit fields for generative shape modeling. In *Proceedings of the IEEE/CVF Conference on Computer Vision and Pattern Recognition*, pages 5939–5948, 2019.

[14] Lars Mescheder, Michael Oechsle, Michael Niemeyer, Sebastian Nowozin, and Andreas Geiger. Occupancy networks: Learning 3d reconstruction in function space. In *Proceedings of the IEEE/CVF Conference on Computer Vision and Pattern Recognition*, pages 4460–4470, 2019.

[15] Yuanhui Huang, Wenzhao Zheng, Yunpeng Zhang, Jie Zhou, and Jiwen Lu. Tri-perspective view for vision-based 3d semantic occupancy prediction. In *Proceedings of the IEEE/CVF Conference on Computer Vision and Pattern Recognition*, pages 9223–9232, 2023.

[16] Yiming Li, Zhiding Yu, Christopher Choy, Chaowei Xiao, Jose M Alvarez, Sanja Fidler, Chen Feng, and Anima Anandkumar. Voxformer: Sparse voxel transformer for camera-based 3d semantic scene completion. In *Proceedings of the IEEE/CVF Conference on Computer Vision and Pattern Recognition*, pages 9087–9098, 2023.

[17] Yi Wei, Linqing Zhao, Wenzhao Zheng, Zheng Zhu, Jie Zhou, and Jiwen Lu. Surroundocc: Multi-camera 3d occupancy prediction for autonomous driving. In *Proceedings of the IEEE/CVF International Conference on Computer Vision*, pages 21729–21740, 2023.

[18] Yunpeng Zhang, Zheng Zhu, and Dalong Du. Occformer: Dual-path transformer for vision-based 3d semantic occupancy prediction. In *Proceedings of the IEEE/CVF International Conference on Computer Vision*, pages 9433–9443, 2023.

[19] Songyou Peng, Michael Niemeyer, Lars Mescheder, Marc Pollefeys, and Andreas Geiger. Convolutional occupancy networks. In *Computer Vision–ECCV 2020: 16th European Conference, Glasgow, UK, August 23–28, 2020, Proceedings, Part III 16*, pages 523–540. Springer, 2020.

[20] Zhen Wang, Shijie Zhou, Jeong Joon Park, Despoina Paschalidou, Suya You, Gordon Wetzstein, Leonidas Guibas, and Achuta Kadambi. Alto: Alternating latent topologies for implicit 3d reconstruction. In *Proceedings of the IEEE/CVF Conference on Computer Vision and Pattern Recognition*, pages 259–270, 2023.

[21] Thomas Müller, Alex Evans, Christoph Schied, and Alexander Keller. Instant neural graphics primitives with a multiresolution hash encoding. *ACM transactions on graphics (TOG)*, 41(4):1–15, 2022.

[22] Jiahui Huang, Zan Gojcic, Matan Atzmon, Or Litany, Sanja Fidler, and Francis Williams. Neural kernel surface reconstruction. In *Proceedings of the IEEE/CVF Conference on Computer Vision and Pattern Recognition*, pages 4369–4379, 2023.

[23] Jongmin Lee, Byungjin Kim, Seungwook Kim, and Minsu Cho. Learning rotation-equivariant features for visual correspondence. In *Proceedings of the IEEE/CVF Conference on Computer Vision and Pattern Recognition*, pages 21887–21897, 2023.

[24] Yunlu Chen, Basura Fernando, Hakan Bilen, Matthias Nießner, and Efstratios Gavves. 3d equivariant graph implicit functions. In *European Conference on Computer Vision*, pages 485–502. Springer, 2022.

[25] Ruixuan Yu and Jian Sun. Pose-transformed equivariant network for 3d point trajectory prediction. In *Proceedings of the IEEE/CVF Conference on Computer Vision and Pattern Recognition*, 2024.

[26] Yiyang Chen, Lunhao Duan, Shanshan Zhao, Changxing Ding, and Dacheng Tao. Local-consistent transformation learning for rotation-invariant point cloud analysis. In *Proceedings of the IEEE/CVF Conference on Computer Vision and Pattern Recognition*, 2024.

[27] Congyue Deng, Or Litany, Yueqi Duan, Adrien Poulenard, Andrea Tagliasacchi, and Leonidas J Guibas. Vector neurons: A general framework for so (3)-equivariant networks. In *Proceedings of the IEEE/CVF International Conference on Computer Vision*, pages 12200–12209, 2021.

[28] Chunghyun Park, Seungwook Kim, Jaesik Park, and Minsu Cho. Learning so(3)-invariant correspondence via point-wise local shape transform. In *Proceedings of the IEEE/CVF Conference on Computer Vision and Pattern Recognition*, 2024.

[29] Alexandre Agm Duval, Victor Schmidt, Alex Hernández-García, Santiago Miret, Fragkiskos D Malliaros, Yoshua Bengio, and David Rolnick. Faenet: Frame averaging equivariant gnn for materials modeling. In *International Conference on Machine Learning*, pages 9013–9033. PMLR, 2023.

[30] Omri Puny, Matan Atzmon, Edward J Smith, Ishan Misra, Aditya Grover, Heli Ben-Hamu, and Yaron Lipman. Frame averaging for invariant and equivariant network design. In *International Conference on Learning Representations*, 2021.

[31] Carlos Esteves, Christine Allen-Blanchette, Ameesh Makadia, and Kostas Daniilidis. Learning so (3) equivariant representations with spherical cnns. In *Proceedings of the European Conference on Computer Vision (ECCV)*, pages 52–68, 2018.

[32] Nathaniel Thomas, Tess Smidt, Steven Kearnes, Lusann Yang, Li Li, Kai Kohlhoff, and Patrick Riley. Tensor field networks: Rotation-and translation-equivariant neural networks for 3d point clouds. *arXiv preprint arXiv:1802.08219*, 2018.

[33] Fabian Fuchs, Daniel Worrall, Volker Fischer, and Max Welling. Se (3)-transformers: 3d roto-translation equivariant attention networks. *Advances in Neural Information Processing Systems*, 33:1970–1981, 2020.

[34] Víctor Garcia Satorras, Emiel Hoogeboom, and Max Welling. E (n) equivariant graph neural networks. In *International Conference on Machine Learning*, pages 9323–9332, 2021.

[35] Wenbing Huang, Jiaqi Han, Yu Rong, Tingyang Xu, Fuchun Sun, and Junzhou Huang. Equivariant graph mechanics networks with constraints. In *International Conference on Learning Representations*, 2021.

[36] Kristof Schütt, Oliver Unke, and Michael Gastegger. Equivariant message passing for the prediction of tensorial properties and molecular spectra. In *International Conference on Machine Learning*, pages 9377–9388, 2021.

[37] Fang Wu and Stan Z Li. Diffmd: a geometric diffusion model for molecular dynamics simulations. In *Proceedings of the AAAI Conference on Artificial Intelligence*, volume 37, pages 5321–5329, 2023.

[38] Yangtian Zhang, Huiyu Cai, Chence Shi, and Jian Tang. E3bind: An end-to-end equivariant network for protein-ligand docking. In *The Eleventh International Conference on Learning Representations*, 2022.

[39] Chen Chen, Xiao Chen, Alex Morehead, Tianqi Wu, and Jianlin Cheng. 3d-equivariant graph neural networks for protein model quality assessment. *Bioinformatics*, 39(1):btad030, 2023.

[40] Rahmatullah Roche, Bernard Moussad, Md Hossain Shuvo, and Debswapna Bhattacharya. E (3) equivariant graph neural networks for robust and accurate protein-protein interaction site prediction. *PLoS Computational Biology*, 19(8):e1011435, 2023.

[41] Haojie Huang, Dian Wang, Robin Walters, and Robert Platt. Equivariant transporter network. In *Robotics: Science and Systems*, 2022.

[42] Dian Wang, Mingxi Jia, Xupeng Zhu, Robin Walters, and Robert Platt. On-robot learning with equivariant models. In *Conference on Robot Learning*, pages 1345–1354. PMLR, 2023.

[43] Xupeng Zhu, Dian Wang, Guanang Su, Ondrej Biza, Robin Walters, and Robert Platt. On robot grasp learning using equivariant models. *Autonomous Robots*, 47(8):1175–1193, 2023.

[44] Jiahui Lei, Congyue Deng, Karl Schmeckpeper, Leonidas Guibas, and Kostas Daniilidis. Efem: Equivariant neural field expectation maximization for 3d object segmentation without scene supervision. In *Proceedings of the IEEE/CVF Conference on Computer Vision and Pattern Recognition*, pages 4902–4912, 2023.

[45] Gene H Golub and Christian Reinsch. Singular value decomposition and least squares solutions. In *Handbook for Automatic Computation: Volume II: Linear Algebra*, pages 134–151. Springer, 1971.

[46] Chen Zhao, Jiaqi Yang, Xin Xiong, Angfan Zhu, Zhiguo Cao, and Xin Li. Rotation invariant point cloud analysis: Where local geometry meets global topology. *Pattern Recognition*, 127:108626, 2022.

[47] Hao Wang, Yitong Wang, Zheng Zhou, Xing Ji, Dihong Gong, Jingchao Zhou, Zhifeng Li, and Wei Liu. Cosface: Large margin cosine loss for deep face recognition. In *Proceedings of the IEEE/CVF Conference on Computer Vision and Pattern Recognition*, pages 5265–5274, 2018.

[48] Benoit Guillard, Federico Stella, and Pascal Fua. Meshudf: Fast and differentiable meshing of unsigned distance field networks. In *European Conference on Computer Vision*, pages 576–592. Springer, 2022.

[49] Adam Paszke, Sam Gross, Francisco Massa, Adam Lerer, James Bradbury, Gregory Chanan, Trevor Killeen, Zeming Lin, Natalia Gimelshein, Luca Antiga, et al. Pytorch: An imperative style, high-performance deep learning library. *Advances in Neural Information Processing Systems*, 32, 2019.

[50] Diederik P Kingma and Jimmy Ba. Adam: A method for stochastic optimization. *arXiv preprint arXiv:1412.6980*, 2014.

[51] Angel X Chang, Thomas Funkhouser, Leonidas Guibas, Pat Hanrahan, Qixing Huang, Zimo Li, Silvio Savarese, Manolis Savva, Shuran Song, Hao Su, et al. Shapenet: An information-rich 3d model repository. *arXiv preprint arXiv:1512.03012*, 2015.

[52] Sebastian Koch, Albert Matveev, Zhongshi Jiang, Francis Williams, Alexey Artemov, Evgeny Burnaev, Marc Alexa, Denis Zorin, and Daniele Panozzo. Abc: A big cad model dataset for geometric deep learning. In *Proceedings of the IEEE/CVF Conference on Computer Vision and Pattern Recognition*, pages 9601–9611, 2019.

[53] Bharat Lal Bhatnagar, Garvita Tiwari, Christian Theobalt, and Gerard Pons-Moll. Multi-garment net: Learning to dress 3d people from images. In *Proceedings of the IEEE/CVF International Conference on Computer Vision*, pages 5420–5430, 2019.

[54] Yizhi Wang, Zeyu Huang, Ariel Shamir, Hui Huang, Hao Zhang, and Ruizhen Hu. Aro-net: Learning implicit fields from anchored radial observations. In *Proceedings of the IEEE/CVF Conference on Computer Vision and Pattern Recognition*, pages 3572–3581, 2023.

[55] Jingwei Huang, Hao Su, and Leonidas Guibas. Robust watertight manifold surface generation method for shapenet models. *arXiv preprint arXiv:1802.01698*, 2018.

# Appendix

## A  Proofs

### A.1  Proof of Lemma 1

**Lemma 1.** *With $\tau$ as our pose normalization, and $D$ as displacement predictor, $D \circ \{\tau(P),\ \tau(q)\}$ is invariant under $SE(3)$ transformation of $P$.*

Assume that $P,\ Y \in \mathbb{R}^{K \times 3}$ are two point patches with $Y = \zeta(P)$, $\zeta \in SE(3)$, and $\zeta(P) = PR + T$ is the $SE(3)$ transformation of $P$ with rotation matrix $R$ and translation vector $T$. Denote the patch centers of $P, Y$ are $\mu, \nu$ respectively, and their corresponding PCA-normalization are $\tau_P(P) = (P - \mu)U$, $\tau_Y(Y) = (Y - \nu)V$. To prove Lemma 1, we firstly prove the uniqueness of pose-normalization, then prove the $SE(3)$-invariance of $D \circ \{\tau(P),\ \tau(q)\}$.

**Step 1.** *Uniqueness of PCA-normalization $\tau$.* According to the definition of $\tau_P(P) = (P - \mu)U$, the patch center $\mu$ is uniquely computed as patch centroid, while rotation matrix $U$ is computed by SVD over the covariance matrix $(P - \mu)^\top(P - \mu)$. The vector elements of matrix $U = \{u_i\}_{i=1}^3$ may change their directions and result in eight rotation matrix $\hat{U} = \{\pm u_i\}_{i=1}^3$, which bring uncertainty for pose-normalization $\tau_P$. To uniquely determine $U$, we follow [46] to determine a single direction for every $\{u_i\}_{i=1}^3$ by estimating their angle with a predefined anchor point $y$ (the vector from the farthest point of the patch to the patch center). The direction of $u_i$ should be flipped if the corresponding angle is larger than $90°$. Specifically, if $\langle u_i,\ y \rangle > 0$, we take $u_i$ as one vector of $U$. If $\langle u_i,\ y \rangle < 0$, we take $-u_i$ instead. If $\langle u_i,\ y \rangle = 0$, we take another point $y'$ (e.g., the second farthest point from patch center) that satisfies $\langle u_i,\ y' \rangle \neq 0$ as a new anchor point to determine the $U$. By this strategy, the rotation matrix $U$ is uniquely determined and the PCA-normalized $\tau_P(P)$ is uniquely determined.

**Step 2.** *$SE(3)$-invariance of $D \circ \{\tau_P(P),\ \tau_P(q)\}$.* For point patch $P = \{p_i\}$ and its $SE(3)$ transformed patch $Y = \{y_i | \ y_i = p_i R + T\}$, with their corresponding centers $\mu, \nu$ computed by

$$\mu = \frac{1}{K}\sum_{i=1}^{K} p_i, \quad \nu = \frac{1}{K}\sum_{i=1}^{K} y_i, \tag{17}$$

we have

$$\nu = \frac{1}{K}\sum_{i=1}^{K} y_i = \frac{1}{K}\sum_{i=1}^{K}(p_i R + T) = \left(\frac{1}{K}\sum_{i=1}^{K} p_i\right) R + T = \mu R + T. \tag{18}$$

Then it is obvious

$$Y - \nu = PR + T - (\mu R + T) = PR - \mu R = (P - \mu)R, \tag{19}$$

which means $Y - \nu$ is an orthogonal transformation of $P - \mu$, thus $Y - \nu$ has same singular values as $P - \mu$ when we conduct SVD on their corresponding covariance matrices, *i.e.*,

$$
\begin{aligned}
U\Lambda U^\top &= (P - \mu)^\top(P - \mu),\\
V\Lambda V^\top &= (Y - \nu)^\top(Y - \nu)\\
&= [(P - \mu)R]^\top[(P - \mu)R]\\
&= R^\top[(P - \mu)^\top(P - \mu)]R\\
&= R^\top(U\Lambda U^\top)R\\
&= (R^\top U)\Lambda(R^\top U)^\top.
\end{aligned}
\tag{20}
$$

Eqn. (20) holds for any rotation matrix $R$, and recalling that we uniquely conduct SVD as proved in Step 1, we derive

$$V = R^\top U. \tag{21}$$

Finally, according to the definition of our PCA-normalization and Eqns. (19,21), it is obvious that

$$\tau_Y(Y) = (Y - \nu)V = (P - \mu)RR^\top U = (P - \mu)U = \tau_P(P), \tag{22}$$

which means $\tau_P(P)$ is $SE(3)$-invariant. The $SE(3)$-invariance of $\tau_P(q)$ can be proven similarly. Taking the $SE(3)$-invariant $\tau_P(P)$, $\tau_P(q)$ as input, our displacement predictor $D$ will produce $SE(3)$-invariant output, *i.e.*, $D \circ \{\tau_P(P),\ \tau_P(q)\} = D \circ \{\tau_{\zeta(P)}(\zeta(P)),\ \tau_{\zeta(P)}(\zeta(q))\}, \forall \zeta \in SE(3)$.

Table 7: The time to process 10,000 points on the ABC [52] dataset using one NVIDIA 4090 GPU.

| Operator | SVD | PE | SRM | PFEM | IPGE | Others | Total |
|----------|-----|-----|-----|------|------|--------|-------|
| Time(s) | 0.4473 | 0.1773 | 0.0297 | 0.0004 | 0.0008 | 0.02 | 0.6688 |

## A.2   Proof of Theorem 1

**Theorem 1** *Given query point $q$ and patch $P$, implicit function $\mathcal{F}(q)$ is $SE(3)$-equivariant.*

Following the denotations in the proof of Lemma 1, we denote the query point of patch $P$, $Y$ as $p_1, p_2$, and we further denote the predicted $SE(3)$-invariant displacements for query point $p_1, p_2$ as $p_1^*, p_2^*$, and the final prediction of $p_1, p_2$ are respectively

$$\mathcal{F}(p_1) = \tau_{p_1}^{-1}(p_1^*) = p_1^* U^\top + \mu, \quad \mathcal{F}(p_2) = \tau_{p_2}^{-1}(p_2^*) = p_2^* V^\top + \nu. \tag{23}$$

According to Eqns. (19,21), we can derive

$$\begin{aligned}
\mathcal{F}(p_2) &= p_2^* V^\top + \nu \\
&= p_1^* (R^\top U)^\top + (\mu R + T) \\
&= (p_1^* U^\top + \mu) R + T \\
&= \mathcal{F}(p_1) R + T,
\end{aligned} \tag{24}$$

which means $\mathcal{F}(p_1)$ is equivariant under $SE(3)$ transformation of point patch $P$.

## B   Architecture Details

**MLPs.** $\gamma_{\theta_s}$ in Eqn. ( 7) consists of four $1 \times 1$ convolution layers with 6, 32, 64, and 128 hidden units. $\gamma_{\theta_q}$ and $\gamma_{\theta_q}$ in Eqn. (8) consists of $1 \times 1$ convolution layers with 3, 32, 64, and 128 hidden units while these for $\gamma_{\theta_p}$ are 3, 32, 64, and 128. For $\gamma_{\theta_a}$ in Eqn. ( 14), the unit numbers are 256, 512, 256, and 384. For $\gamma_{\theta_d}$, the unit numbers are 256, 256, 256, and 256. All feature dimensions are 128. For the multi-head memory bank $\mathcal{M}$, the number of the memory bank is set as $N_M = 4$, with each memory bank containing 596 items.

## C   Additional Results

### C.1   The visualization of the learned memory bank

In Figure 7, we provided two approaches to visualize the learned memory bank.

(1) We visualize the set of point patches with the highest weights to the corresponding element of the memory bank (the weights are computed by Eqn. (12). These patches are highlighted by colors in these examples. It shows that the patches with high weights to each element of memory have similar geometry structures.

(2) We further visualize (by t-SNE) the features of point patches with the highest weights (Eqn. (12)) to different elements of the learned memory bank, rendered by different colors. It shows that the patches with high weights assigned to different elements of the learned memory bank have clustered features in the feature space.

### C.2   A detailed analysis of the inference time

In Table 7, we report the time consumption of each operator in PEIF to process 10,000 query points. Specifically, the operations include SVD (Singular Value Decomposition), PE (Point-wise Feature Extraction), SRM (Spatial Relation Module), PFEM (Patch Feature Extraction Module), IPGE (Intrinsic Patch Geometry Extractor) and Others (other Conv layers).

### C.3   Computational cost

We report the computational cost in Table 8, including the training time per epoch, training memory, testing time per 3D shape, and testing memory cost on the ABC dataset. Methods of GeoUDF and

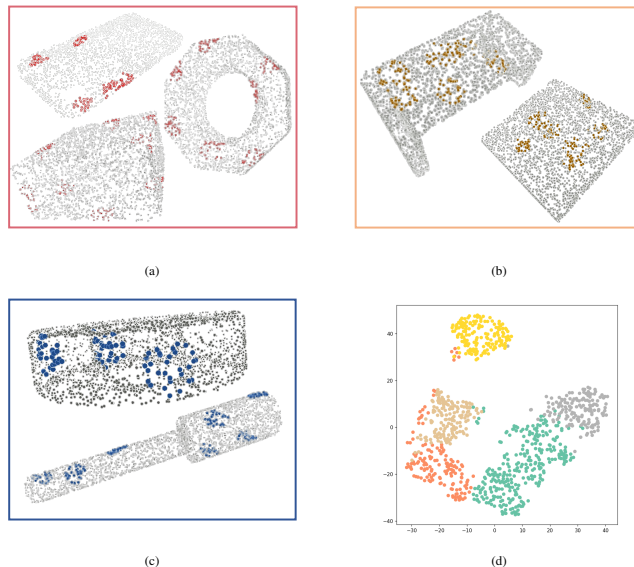

Figure 7: The visualization of the learned memory bank. (a)-(c) Visualization of patches corresponding to the same memory item with the highest weights (Different colors represent different items). (d) t-SNE visualization of patch features. The patches with the highest weights to different memory items are highlighted in different colors.

Table 8: The comparison of computational cost.

| Cost | Training Time (s) | Training Mem (G) | Testing Time (s) | Testing Mem (G) |
|---|---|---|---|---|
| NVF | 28.06 | 6.70 | 73.9 | 0.66 |
| GeoUDF | 61.20+58.78 | 14.99+14.99 | 124.73 | 2.27 |
| GridFormer | 26.48+26.78 | 6.59+6.59 | 13.32 | 0.31 |
| E-GraphONet | 37.82 | 16.11 | 1.92 | 1.49 |
| PEIF (Ours) | 34.56 | 17.75 | 40.98 | 1.56 |

GridFormer include two stages of upsampling/reconstruction and reconstruction/refinement. We report the computation cost of them in each table cell with two values (denoted as $\cdot + \cdot$), respectively representing the costs for each stage.

## C.4 Additional results on degraded data

We evaluate the performance of our PEIF on different data degradations (sparse, noisy, and partial) on the ABC [52] dataset. In the following experiments, the test input point clouds with different degradations, and the results are reported in Tables 9-11.

**Sparse point cloud.** In previous experiments, all the compared methods use the same number of input points (10k) for each shape in testing, as NVF. We randomly select a subset of input points as input, and the results are in Table 9.

Table 9: The reconstruction results of sparse data on the ABC [52] dataset.

| Method | $N = 5k$ | | | | $N = 2k$ | | | |
|---|---|---|---|---|---|---|---|---|
| | CD ↓ | EMD ↓ | NC ↑ | F-Score ↑ | CD ↓ | EMD ↓ | NC ↑ | F-Score ↑ |
| NVF [9] | 0.297 | 2.706 | 0.935 | 0.979 | 0.409 | 2.725 | 0.932 | 0.946 |
| GeoUDF [7] | 0.306 | 2.726 | 0.940 | 0.985 | 0.399 | 2.711 | 0.935 | 0.952 |
| GridFormer [11] | 0.292 | 2.694 | **0.952** | 0.982 | 0.369 | 2.703 | **0.945** | 0.956 |
| PEIF(Ours) | **0.269** | **2.679** | 0.945 | **0.988** | **0.360** | **2.685** | 0.938 | **0.960** |

**Noisy point cloud.** We plugged Gaussian noise with standard deviation ($\sigma$) as 0.005 and 0.01 to the input points. The results are reported in Table 10.

Table 10: The reconstruction results from noisy input on the ABC [52] dataset.

| Method | $\sigma = 0.005$ | | | | $\sigma = 0.01$ | | | |
|---|---|---|---|---|---|---|---|---|
| | CD ↓ | EMD ↓ | NC ↑ | F-Score ↑ | CD ↓ | EMD ↓ | NC ↑ | F-Score ↑ |
| NVF [9] | 0.512 | 3.257 | 0.712 | 0.924 | 0.792 | 3.687 | 0.723 | 0.693 |
| GeoUDF [7] | 0.496 | 3.268 | 0.732 | 0.911 | 0.785 | 3.428 | 0.710 | 0.655 |
| GridFormer [11] | 0.839 | 3.321 | **0.793** | 0.805 | 1.132 | 3.379 | **0.759** | 0.510 |
| PEIF(Ours) | **0.480** | **3.132** | 0.745 | **0.952** | **0.773** | **3.358** | 0.715 | **0.702** |

**Partial point cloud.** We remove a fraction (with ratio $p$) of the input points to form a partial point cloud. Specifically, we use the farthest point sampling to select a set of center points and remove their KNN points to ensure the sampling fraction. The results are reported in Table 11.

Table 11: The reconstruction results from partial points on the ABC [52] dataset.

| Method | $p = 10\%$ | | | | $p = 20\%$ | | | |
|---|---|---|---|---|---|---|---|---|
| | CD ↓ | EMD ↓ | NC ↑ | F-Score ↑ | CD ↓ | EMD ↓ | NC ↑ | F-Score ↑ |
| NVF [9] | 0.264 | 2.697 | 0.943 | 0.992 | 0.274 | 2.710 | 0.940 | 0.990 |
| GeoUDF [7] | 0.268 | 2.695 | **0.959** | 0.994 | 0.275 | 2.745 | 0.947 | 0.991 |
| GridFormer [11] | 0.267 | 2.706 | 0.964 | 0.982 | 0.298 | 2.746 | 0.946 | 0.987 |
| PEIF(Ours) | **0.246** | **2.692** | 0.960 | **0.996** | **0.249** | **2.697** | **0.956** | **0.995** |

## C.5 Additional ablation study

**Ablation study in ABC [52].** In Table 13, we present the additional results of our ablation studies with respect to the coefficient $\beta$ of the joint loss function in Section 4.4, the impact of keypoint-boosted feature representation $f_{\bar{P}_s}$ in Section 4.3.

**Ablation study in MGN [53].** In Table 12, we present the additional ablation results of $K$ on MGN dataset. The testing results in Table 3 on the MNG dataset using the trained model with $K = 54$ on the Synthetic Rooms dataset. As shown in Table 12, when changing $K$ to 48 and 32, the test results using the corresponding $K$ on MGN are stable.

Table 12: The impact of $K$ when training on Synthetic Rooms and testing on MGN [53].

| $K$ | CD ↓ | EMD ↓ | NC ↑ | F-Score ↑ |
|---|---|---|---|---|
| 48 | 0.247 | 2.724 | 0.961 | 0.998 |
| 32 | 0.252 | 2.735 | 0.959 | 0.991 |

## C.6 Additional Visualizations

In this section, we provide more visual results of our PEIF across four datasets: ShapeNet, ABC, Synthetic Rooms, and MGN.

**ShapeNet [51].** We present three examples to further illustrate the performance of our PEIF. The 1st row in Figure 8 are results of examples selected from base classes. In contrast, results in the 2nd and 3rd rows come from the novel classes in this dataset. The visualization results show that our PEIF not only preserves local structures but also demonstrates a remarkable generalization capability.

**ABC [52].** We validate the efficacy of our multi-head memory bank on this dataset, with visualizations depicted as shown in Figure 9. By using the intrinsic patch geometry extractor with a multi-head memory bank, our PEIF reconstructs better 3D surfaces which are more smooth and complete.

**Synthetic Rooms [19].** Beyond the reconstruction of CAD objects, we also validate the performance of our PEIF on a synthesized scene dataset, thereby extending its applicability to more complex environmental contexts. The results in Figure 10 demonstrate that our PEIF achieves competitive results in terms of structural completeness and surface smoothness.

Table 13: Additional ablation study in ABC [52].

| | Setting | CD ↓ | EMD ↓ | NC ↑ | F-Score ↑ |
|---|---|---|---|---|---|
| Feature $f_{\bar{P}_s}$ | w/o | 0.247 | 2.684 | 0.961 | 0.997 |
| | w/ | 0.241 | 2.672 | 0.969 | 0.998 |
| $\beta$ | $\beta = 0.0$ | 0.244 | 2.705 | 0.962 | 0.997 |
| | $\beta = 0.01$ | 0.245 | 2.683 | 0.960 | 0.996 |
| | $\beta = 0.1$ | 0.241 | 2.672 | 0.969 | 0.998 |
| | $\beta = 0.5$ | 0.248 | 2.732 | 0.961 | 0.996 |

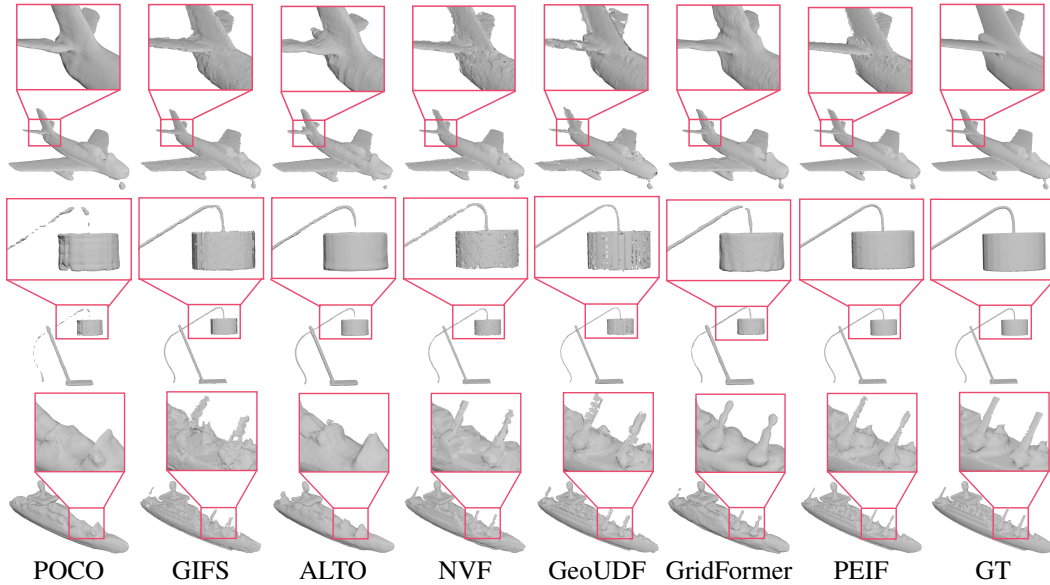

| POCO | GIFS | ALTO | NVF | GeoUDF | GridFormer | PEIF | GT |

Figure 8: Results on ShapeNet [51] dataset. The results from the 1st row are selected from base classes in Table 1. Objects in the 2nd to 3rd rows are selected from meshes used for class-unseen reconstruction (novel classes in Table 1).

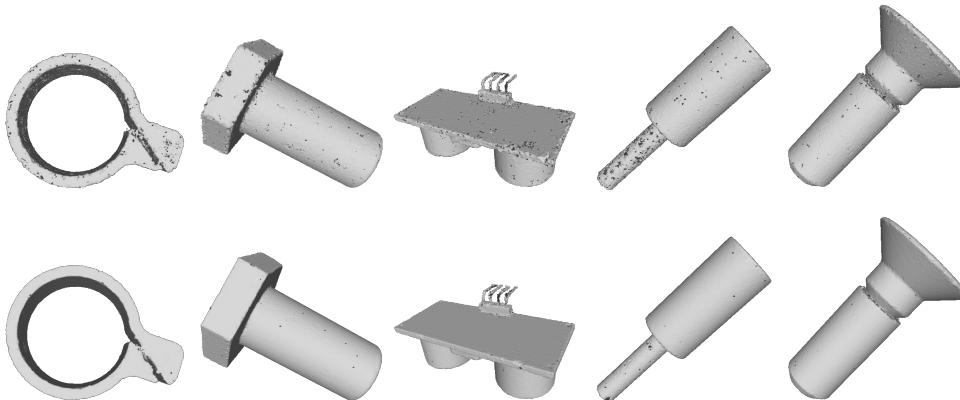

Figure 9: Examples of results on ABC [52] dataset with (below) and without (top) intrinsic patch geometry extractor using multi-head memory bank.

**MGN [53].** To evaluate the generalization of competing methods, we conducted tests on the real scanned data, and the corresponding results are presented in Figure 11. The results indicate that our PEIF yields reconstruction results with better completeness and surface smoothness.

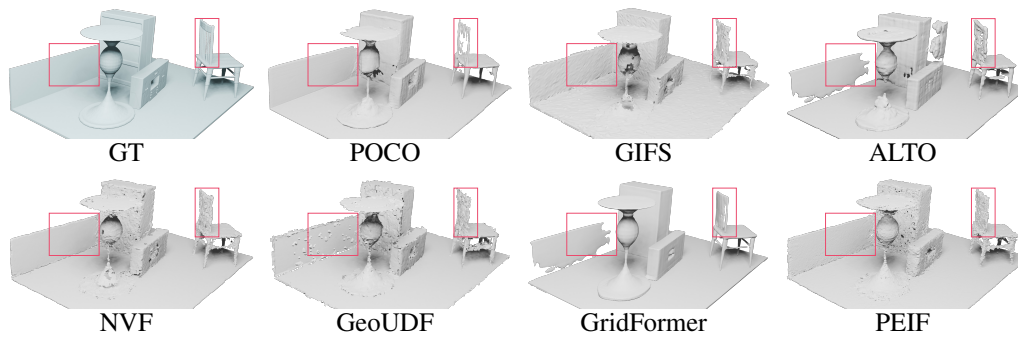

Figure 10: Examples of results on Synthetic Room dataset.

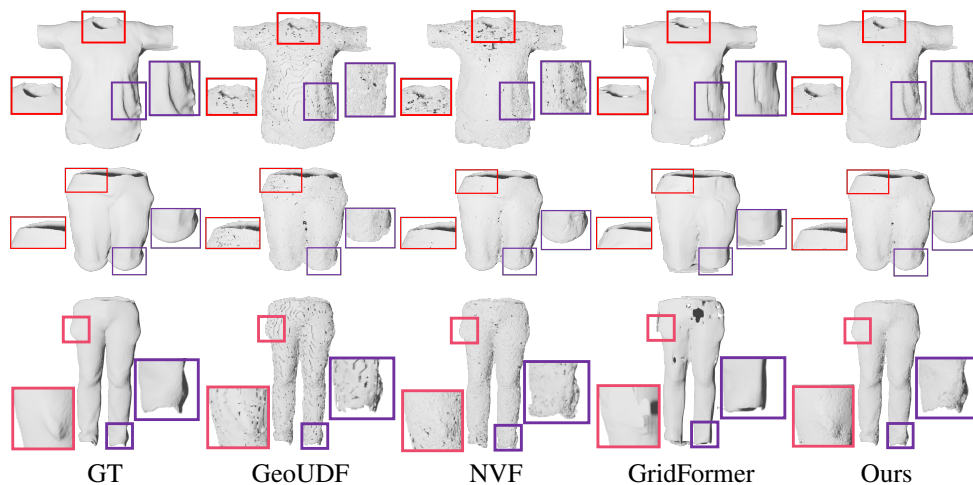

Figure 11: The visual example of cross-domain evaluation on the real scanned dataset MGN [53], where the model is trained on Synthetic Room dataset [19].

